# A New Learning Algorithm for Blind Signal Separation

**S. Amari***
University of Tokyo
Bunkyo-ku, Tokyo 113, JAPAN
amari@sat.t.u-tokyo.ac.jp

**A. Cichocki**
Lab. for Artificial Brain Systems
FRP, RIKEN
Wako-Shi, Saitama, 351-01, JAPAN
cia@kamo.riken.go.jp

**H. H. Yang**
Lab. for Information Representation
FRP, RIKEN
Wako-Shi, Saitama, 351-01, JAPAN
hhy@koala.riken.go.jp

## Abstract

A new on-line learning algorithm which minimizes a statistical dependency among outputs is derived for blind separation of mixed signals. The dependency is measured by the average mutual information (MI) of the outputs. The source signals and the mixing matrix are unknown except for the number of the sources. The Gram-Charlier expansion instead of the Edgeworth expansion is used in evaluating the MI. The natural gradient approach is used to minimize the MI. A novel activation function is proposed for the on-line learning algorithm which has an equivariant property and is easily implemented on a neural network like model. The validity of the new learning algorithm are verified by computer simulations.

## 1   INTRODUCTION

The problem of blind signal separation arises in many areas such as speech recognition, data communication, sensor signal processing, and medical science. Several neural network algorithms [3, 5, 7] have been proposed for solving this problem. The performance of these algorithms is usually affected by the selection of the activation functions for the formal neurons in the networks. However, all activation

functions attempted are monotonic and the selections of the activation functions
are ad hoc. How should the activation function be determined to minimize the MI?
Is it necessary to use monotonic activation functions for blind signal separation? In
this paper, we shall answer these questions and give an on-line learning algorithm
which uses a non-monotonic activation function selected by the independent com-
ponent analysis (ICA) [7]. Moreover, we shall show a rigorous way to derive the
learning algorithm which has the equivariant property, i.e., the performance of the
algorithm is independent of the scaling parameters in the noiseless case.

## 2   PROBLEM

Let us consider unknown source signals $s^i(t), i = 1, \cdots, n$ which are mutually in-
dependent. It is assumed that the sources $s^i(t)$ are stationary processes and each
source has moments of any order with a zero mean. The model for the sensor output
is

$$\mathbf{x}(t) = \boldsymbol{A}\mathbf{s}(t)$$

where $\boldsymbol{A} \in \boldsymbol{R}^{n \times n}$ is an unknown non-singular mixing matrix, $\mathbf{s}(t) = [s^1(t), \cdots, s^n(t)]^T$ and $\mathbf{x}(t) = [x^1(t), \cdots, x^n(t)]^T$.

Without knowing the source signals and the mixing matrix, we want to recover the
original signals from the observations $\mathbf{x}(t)$ by the following linear transform:

$$\mathbf{y}(t) = \boldsymbol{W}\mathbf{x}(t)$$

where $\mathbf{y}(t) = [y^1(t), \cdots, y^n(t)]^T$ and $\boldsymbol{W} \in \boldsymbol{R}^{n \times n}$ is a de-mixing matrix.

It is impossible to obtain the original sources $s^i(t)$ because they are not identifiable
in the statistical sense. However, except for a permutation of indices, it is possible
to obtain $c_i s^i(t)$ where the constants $c_i$ are indefinite nonzero scalar factors. The
source signals are identifiable in this sense. So our goal is to find the matrix $\boldsymbol{W}$ such
that $[y^1, \cdots, y^n]$ coincides with a permutation of $[s^1, \cdots, s^n]$ except for the scalar
factors. The solution $\boldsymbol{W}$ is the matrix which finds all independent components in
the outputs. An on-line learning algorithm for $\boldsymbol{W}$ is needed which performs the
ICA. It is possible to find such a learning algorithm which minimizes the dependency
among the outputs. The algorithm in [6] is based on the Edgeworth expansion[8] for
evaluating the marginal negentropy. Both the Gram-Charlier expansion[8] and the
Edgeworth expansion[8] can be used to approximate probability density functions.
We shall use the Gram-Charlier expansion instead of the Edgeworth expansion for
evaluating the marginal entropy. We shall explain the reason in section 3.

## 3   INDEPENDENCE OF SIGNALS

The mathematical framework for the ICA is formulated in [6]. The basic idea of the
ICA is to minimize the dependency among the output components. The dependency
is measured by the Kullback-Leibler divergence between the joint and the product
of the marginal distributions of the outputs:

$$D(\boldsymbol{W}) = \int p(\boldsymbol{y}) \log \frac{p(\boldsymbol{y})}{\prod_{a=1}^{n} p_a(y^a)} d\boldsymbol{y} \qquad (1)$$

where $p_a(y^a)$ is the marginal probability density function (pdf). Note the Kullback-
Leibler divergence has some invariant properties from the differential-geometrical
point of view[1].

It is easy to relate the Kullback-Leibler divergence $D(\boldsymbol{W})$ to the average MI of y:

$$D(\boldsymbol{W}) = -H(\mathbf{y}) + \sum_{a=1}^{n} H(y^a) \qquad (2)$$

where

$H(\mathbf{y}) = -\int p(\boldsymbol{y}) \log p(\boldsymbol{y}) d\boldsymbol{y},$
$H(y^a) = -\int p_a(y^a) \log p_a(y^a) dy^a$ is the marginal entropy.

The minimization of the Kullback-Leibler divergence leads to an ICA algorithm for estimating $\boldsymbol{W}$ in [6] where the Edgeworth expansion is used to evaluate the negentropy. We use the truncated Gram-Charlier expansion to evaluate the Kullback-Leibler divergence. The Edgeworth expansion has some advantages over the Gram-Charlier expansion only for some special distributions. In the case of the Gamma distribution or the distribution of a random variable which is the sum of iid random variables, the coefficients of the Edgeworth expansion decrease uniformly. However, there is no such advantage for the mixed output $y^a$ in general cases.

To calculate each $H(y^a)$ in (2), we shall apply the Gram-Charlier expansion to approximate the pdf $p_a(y^a)$. Since $E[\mathbf{y}] = E[\boldsymbol{W}\boldsymbol{A}\mathbf{s}] = 0$, we have $E[y^a] = 0$. To simplify the calculations for the entropy $H(y^a)$ to be carried out later, we assume $m_2^a = 1$. We use the following truncated Gram-Charlier expansion to approximate the pdf $p_a(y^a)$:

$$p_a(y^a) \approx \alpha(y^a)\{1 + \frac{\kappa_3^a}{3!} H_3(y^a) + \frac{\kappa_4^a}{4!} H_4(y^a)\} \qquad (3)$$

where $\kappa_3^a = m_3^a$, $\kappa_4^a = m_4^a - 3$, $m_k^a = E[(y^a)^k]$ is the k-th order moment of $y^a$, $\alpha(y) = \frac{1}{\sqrt{2\pi}} e^{-\frac{y^2}{2}}$, and $H_k(y)$ are Chebyshev-Hermite polynomials defined by the identity

$$(-1)^k \frac{d^k \alpha(y)}{dy^k} = H_k(y)\alpha(y).$$

We prefer the Gram-Charlier expansion to the Edgeworth expansion because the former clearly shows how $\kappa_3^a$ and $\kappa_4^a$ affect the approximation of the pdf. The last term in (3) characterizes non-Gaussian distributions. To apply (3) to calculate $H(y^a)$, we need the following integrals:

$$-\int \alpha(y) H_2(y) \log \alpha(y) dy = \frac{1}{4} \qquad (4)$$

$$\int \alpha(y)(H_2(y))^2 H_4(y) dy = 24. \qquad (5)$$

These integrals can be obtained easily from the following results for the moments of a Gaussian random variable N(0,1):

$$\int y^{2k+1} \alpha(y) dy = 0, \quad \int y^{2k} \alpha(y) dy = 1 \cdot 3 \cdots (2k-1). \qquad (6)$$

By using the expansion

$$\log(1+y) \approx y - \frac{y^2}{2} + O(y^3)$$

and taking account of the orthogonality relations of the Chebyshev-Hermite polynomials and (4)-(5), the entropy $H(y^a)$ is expanded as

$$H(y^a) \approx \frac{1}{2} \log(2\pi e) - \frac{(\kappa_3^a)^2}{2 \cdot 3!} - \frac{(\kappa_4^a)^2}{2 \cdot 4!} + \frac{5}{8}(\kappa_3^a)^2 \kappa_4^a + \frac{1}{16}(\kappa_4^a)^3. \qquad (7)$$

It is easy to calculate

$$-\int \alpha(y)\log \alpha(y)dy = \frac{1}{2}\log(2\pi e).$$

From $\mathbf{y} = \boldsymbol{W}\mathbf{x}$, we have $H(\mathbf{y}) = H(\mathbf{x}) + \log|det(\boldsymbol{W})|$. Applying (7) and the above expressions to (2), we have

$$
\begin{aligned}
D(\boldsymbol{W}) \approx \ & -H(\mathbf{x}) - \log|det(\boldsymbol{W})| + \frac{n}{2}\log(2\pi e) - \sum_{a=1}^{n}[\frac{(\kappa_3^a)^2}{2\cdot 3!} + \frac{(\kappa_4^a)^2}{2\cdot 4!} \\
& -\frac{5}{8}(\kappa_3^a)^2\kappa_4^a - \frac{1}{16}(\kappa_4^a)^3].
\end{aligned}
\tag{8}
$$

## 4   A NEW LEARNING ALGORITHM

To obtain the gradient descent algorithm to update $\boldsymbol{W}$ recursively, we need to calculate $\frac{\partial D}{\partial w_k^a}$ where $w_k^a$ is the (a,k) element of $\boldsymbol{W}$ in the a-th row and k-th column.

Let $cof(w_k^a)$ be the cofactor of $w_k^a$ in $\boldsymbol{W}$. It is not difficult to derive the followings:

$$
\begin{aligned}
\frac{\partial \log|det(\boldsymbol{W})|}{\partial w_k^a} &= \frac{cof(w_k^a)}{det(\boldsymbol{W})} = (\boldsymbol{W}^{-T})_k^a \\
\frac{\partial \kappa_3^a}{\partial w_k^a} &= 3E[(y^a)^2 x^k] \\
\frac{\partial \kappa_4^a}{\partial w_k^a} &= 4E[(y^a)^3 x^k]
\end{aligned}
$$

where $(\boldsymbol{W}^{-T})_k^a$ denotes the (a,k) element of $(\boldsymbol{W}^T)^{-1}$. From (8), we obtain

$$\frac{\partial D}{\partial w_k^a} \approx -(\boldsymbol{W}^{-T})_k^a + f(\kappa_3^a, \kappa_4^a)E[(y^a)^2 x^k] + g(\kappa_3^a, \kappa_4^a)E[(y^a)^3 x^k] \tag{9}$$

where

$$f(y,z) = -\frac{1}{2}y + \frac{15}{4}yz, \quad g(y,z) = -\frac{1}{6}z + \frac{5}{2}y^2 + \frac{3}{4}z^2.$$

From (9), we obtain the gradient descent algorithm to update $\boldsymbol{W}$ recursively:

$$
\begin{aligned}
\frac{dw_k^a}{dt} &= -\eta(t)\frac{\partial D}{\partial w_k^a} \\
&= \eta(t)\{(\boldsymbol{W}^{-T})_k^a - f(\kappa_3^a, \kappa_4^a)E[(y^a)^2 x^k] - g(\kappa_3^a, \kappa_4^a)E[(y^a)^3 x^k]\}
\end{aligned}
\tag{10}
$$

where $\eta(t)$ is a learning rate function. Replacing the expectation values in (10) by their instantaneous values, we have the stochastic gradient descent algorithm:

$$\frac{dw_k^a}{dt} = \eta(t)\{(\boldsymbol{W}^{-T})_k^a - f(\kappa_3^a, \kappa_4^a)(y^a)^2 x^k - g(\kappa_3^a, \kappa_4^a)(y^a)^3 x^k\}. \tag{11}$$

We need to use the following adaptive algorithm to compute $\kappa_3^a$ and $\kappa_4^a$ in (11):

$$
\begin{aligned}
\frac{d\kappa_3^a}{dt} &= -\mu(t)(\kappa_3^a - (y^a)^3) \\
\frac{d\kappa_4^a}{dt} &= -\mu(t)(\kappa_4^a - (y^a)^4 + 3)
\end{aligned}
\tag{12}
$$

where $\mu(t)$ is another learning rate function.

The performance of the algorithm (11) relies on the estimation of the third and fourth order cumulants performed by the algorithm (12). Replacing the moments

of the random variables in (11) by their instantaneous values, we obtain the following algorithm which is a direct but coarse implementation of (11):

$$\frac{dw_k^a}{dt} = \eta(t)\{(\boldsymbol{W}^{-T})_k^a - f(y^a)x^k\} \tag{13}$$

where the activation function $f(y)$ is defined by

$$f(y) = \frac{3}{4}y^{11} + \frac{25}{4}y^9 - \frac{14}{3}y^7 - \frac{47}{4}y^5 + \frac{29}{4}y^3. \tag{14}$$

Note the activation function $f(y)$ is an odd function, not a monotonic function. The equation (13) can be written in a matrix form:

$$\frac{d\boldsymbol{W}}{dt} = \eta(t)\{\boldsymbol{W}^{-T} - \mathbf{f}(\boldsymbol{y})\boldsymbol{x}^T\}. \tag{15}$$

This equation can be further simplified as following by substituting $\mathbf{x}^T\boldsymbol{W}^T = \mathbf{y}^T$:

$$\frac{d\boldsymbol{W}}{dt} = \eta(t)\{\boldsymbol{I} - \mathbf{f}(\boldsymbol{y})\boldsymbol{y}^T\}\boldsymbol{W}^{-T} \tag{16}$$

where $\mathbf{f}(\boldsymbol{y}) = (f(y^1), \cdots, f(y^n))^T$. The above equation is based on the gradient descent algorithm (10) with the following matrix form:

$$\frac{d\boldsymbol{W}}{dt} = -\eta(t)\frac{\partial D}{\partial \boldsymbol{W}}. \tag{17}$$

From information geometry perspective[1], since the mixing matrix $\boldsymbol{A}$ is non-singular we had better replace the above algorithm by the following natural gradient descent algorithm:

$$\frac{d\boldsymbol{W}}{dt} = -\eta(t)\frac{\partial D}{\partial \boldsymbol{W}}\boldsymbol{W}^T\boldsymbol{W}. \tag{18}$$

Applying the previous approximation of the gradient $\frac{\partial D}{\partial \boldsymbol{W}}$ to (18), we obtain the following algorithm:

$$\frac{d\boldsymbol{W}}{dt} = \eta(t)\{\boldsymbol{I} - \mathbf{f}(\boldsymbol{y})\boldsymbol{y}^T\}\boldsymbol{W} \tag{19}$$

which has the same "equivariant" property as the algorithms developed in [4, 5].

Although the on-line learning algorithms (16) and (19) look similar to those in [3, 7] and [5] respectively, the selection of the activation function in this paper is rational, not ad hoc. The activation function (14) is determined by the ICA. It is a non-monotonic activation function different from those used in [3, 5, 7].

There is a simple way to justify the stability of the algorithm (19). Let Vec(·) denote an operator on a matrix which cascades the columns of the matrix from the left to the right and forms a column vector. Note this operator has the following property:

$$\text{Vec}(\boldsymbol{ABC}) = (\boldsymbol{C}^T \otimes \boldsymbol{A})\text{Vec}(\boldsymbol{B}). \tag{20}$$

Both the gradient descent algorithm and the natural gradient descent algorithm are special cases of the following general gradient descent algorithm:

$$\frac{d\text{Vec}(\boldsymbol{W})}{dt} = -\eta(t)\boldsymbol{P}\frac{\partial D}{\partial \text{Vec}(\boldsymbol{W})} \tag{21}$$

where $\boldsymbol{P}$ is a symmetric and positive definite matrix. It is trivial that (21) becomes (17) when $\boldsymbol{P} = \boldsymbol{I}$. When $\boldsymbol{P} = \boldsymbol{W}^T\boldsymbol{W} \otimes \boldsymbol{I}$, applying (20) to (21), we obtain

$$\frac{d\text{Vec}(\boldsymbol{W})}{dt} = -\eta(t)(\boldsymbol{W}^T\boldsymbol{W} \otimes \boldsymbol{I})\frac{\partial D}{\partial \text{Vec}(\boldsymbol{W})} = -\eta(t)\text{Vec}(\frac{\partial D}{\partial \boldsymbol{W}}\boldsymbol{W}^T\boldsymbol{W})$$

and this equation implies (18). So the natural gradient descent algorithm updates $W(t)$ in the direction of decreasing the dependency $D(W)$. The information geometry theory[1] explains why the natural gradient descent algorithm should be used to minimize the MI.

Another on-line learning algorithm for blind separation using recurrent network was proposed in [2]. For this algorithm, the activation function (14) also works well. In practice, other activation functions such as those proposed in [2]-[6] may also be used in (19). However, the performance of the algorithm for such functions usually depends on the distributions of the sources. The activation function (14) works for relatively general cases in which the pdf of each source can be approximated by the truncated Gram-Charlier expansion.

## 5  SIMULATION

In order to check the validity and performance of the new on-line learning algorithm (19), we simulate it on the computer using synthetic source signals and a random mixing matrix. The extensive computer simulations have fully confirmed the theory and the validity of the algorithm (19). Due to the limit of space we present here only one illustrative example.

**Example:**

Assume that the following three unknown sources are mixed by a random mixing matrix $A$:

$$[s^1(t), s^2(t), s^3(t)] = [n(t), 0.1sin(400t)cos(30t), 0.01sign[sin(500t + 9cos(40t))]]$$

where $n(t)$ is a noise source uniformly distributed in the range $[-1, +1]$, and $s_2(t)$ and $s_3(t)$ are two deterministic source signals. The elements of the mixing matrix A are randomly chosen in $[-1, +1]$. The learning rate is exponentially decreasing to zero as $\eta(t) = 250exp(-5t)$.

A simulation result is shown in Figure 1. The first three signals denoted by X1, X2 and X3 represent mixing (sensor) signals: $x^1(t)$, $x^2(t)$ and $x^3(t)$. The last three signals denoted by O1, O2 and O3 represent the output signals: $y^1(t)$, $y^2(t)$, and $y^3(t)$. By using the proposed learning algorithm, the neural network is able to extract the deterministic signals from the observations after approximately 500 milliseconds.

The performance index $E_1$ is defined by

$$E_1 = \sum_{i=1}^{n}(\sum_{j=1}^{n} \frac{|p_{ij}|}{\max_k |p_{ik}|} - 1) + \sum_{j=1}^{n}(\sum_{i=1}^{n} \frac{|p_{ij}|}{\max_k |p_{kj}|} - 1)$$

where $P = (p_{ij}) = WA$.

## 6  CONCLUSION

The major contribution of this paper the rigorous derivation of the effective blind separation algorithm with equivariant property based on the minimization of the MI of the outputs. The ICA is a general principle to design algorithms for blind signal separation. The most difficulties in applying this principle are to evaluate the MI of the outputs and to find a working algorithm which decreases the MI. Different from the work in [6], we use the Gram-Charlier expansion instead of the Edgeworth expansion to calculate the marginal entropy in evaluating the MI. Using

the natural gradient method to minimize the MI, we have found an on-line learning algorithm to find a de-mixing matrix. The algorithm has equivariant property and can be easily implemented on a neural network like model. Our approach provides a rational selection of the activation function for the formal neurons in the network. The algorithm has been simulated for separating unknown source signals mixed by a random mixing matrix. Our theory and the validity of the new learning algorithm are verified by the simulations.

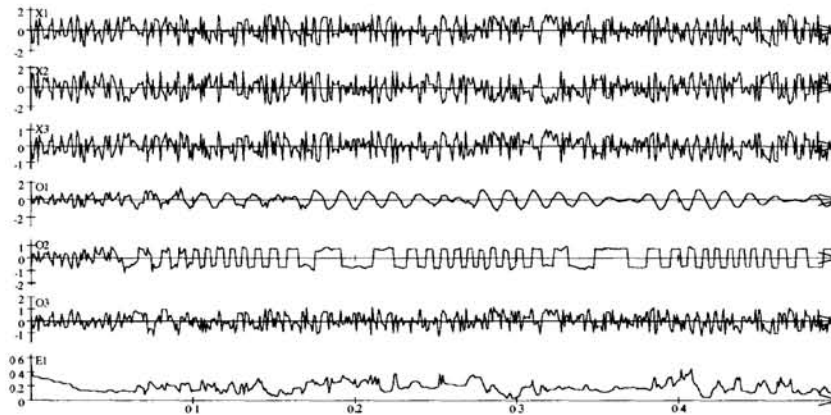

Figure 1: The mixed and separated signals, and the performance index

**Acknowledgment**

We would like to thank Dr. Xiao Yan SU for the proof-reading of the manuscript.

## Footnotes

*Lab. for Information Representation, FRP, RIKEN, Wako-shi, Saitama, JAPAN

# References

[1] S.-I. Amari. *Differential-Geometrical Methods in Statistics, Lecture Notes in Statistics vol.28.* Springer, 1985.

[2] S. Amari, A. Cichocki, and H. H. Yang. Recurrent neural networks for blind separation of sources. In *Proceedings 1995 International Symposium on Nonlinear Theory and Applications*, volume I, pages 37–42, December 1995.

[3] A. J. Bell and T. J. Sejnowski. An information-maximisation approach to blind separation and blind deconvolution. *Neural Computation*, 7:1129–1159, 1995.

[4] J.-F. Cardoso and Beate Laheld. Equivariant adaptive source separation. *To appear in IEEE Trans. on Signal Processing*, 1996.

[5] A. Cichocki, R. Unbehauen, L. Moszczyński, and E. Rummert. A new on-line adaptive learning algorithm for blind separation of source signals. In *ISANN94*, pages 406–411, Taiwan, December 1994.

[6] P. Comon. Independent component analysis, a new concept? *Signal Processing*, 36:287–314, 1994.

[7] C. Jutten and J. Herault. Blind separation of sources, part i: An adaptive algorithm based on neuromimetic architecture. *Signal Processing*, 24:1–10, 1991.

[8] A. Stuart and J. K. Ord. *Kendall's Advanced Theory of Statistics.* Edward Arnold, 1994.
